# Uncertainty, phase and oscillatory hippocampal recall

**Máté Lengyel and Peter Dayan**
Gatsby Computational Neuroscience Unit
University College London
17 Queen Square, London WC1N 3AR, United Kingdom
{lmate,dayan}@gatsby.ucl.ac.uk

## Abstract

Many neural areas, notably, the hippocampus, show structured, dynamical, population behavior such as coordinated oscillations. It has long been observed that such oscillations provide a substrate for representing analog information in the firing *phases* of neurons relative to the underlying population rhythm. However, it has become increasingly clear that it is essential for neural populations to represent *uncertainty* about the information they capture, and the substantial recent work on neural codes for uncertainty has omitted any analysis of oscillatory systems. Here, we observe that, since neurons in an oscillatory network need not only fire once in each cycle (or even at all), uncertainty about the analog quantities each neuron represents by its firing phase might naturally be reported through the degree of concentration of the spikes that it fires. We apply this theory to memory in a model of oscillatory associative recall in hippocampal area CA3. Although it is not well treated in the literature, representing and manipulating uncertainty is fundamental to competent memory; our theory enables us to view CA3 as an effective uncertainty-aware, retrieval system.

## 1   Introduction

In a network such as hippocampal area CA3 that shows prominent oscillations during memory retrieval and other functions [1], there are apparently three, somewhat separate, ways in which neurons might represent information within a single cycle: they must choose how *many* spikes to fire; what the *mean phase* of those spikes is; and how *concentrated* those spikes are about that mean. Most groups working on the theory of spiking oscillatory networks have considered only the second of these – this is true, for instance, of Hopfield's work on olfactory representations [2] and Yoshioka's [3] and Lengyel & Dayan's work [4] on analog associative memories in CA3. Since neurons do really fire more or less than one spike per cycle, and furthermore in a way that can be informationally rich [5, 6], this poses a key question as to what the other dimensions convey.

The number of spikes per cycle is an obvious analog of a conventional firing rate. Recent sophisticated models of firing rates of single neurons and neural populations treat them as representing *uncertainty* about the quantities coded, partly driven by the strong psychophysical and computational evidence that uncertainty plays a key role in many aspects of neural processing [7, 8, 9]. Single neurons can convey the *certainty* of a binary proposition by firing more or less strongly [10, 11]; a whole population can use firing rates to convey uncertainty about a collectively-coded analog quantity [12].

However, if neurons can fire multiple spikes per cycle, then the degree to which the spikes are *concentrated* around a mean phase is an additional channel for representing information. Concentration is not merely an abstract quantity; rather we can expect that the effect of the neuron on its postsynaptic partners will be strongly influenced by the burstiness of the spikes, an effect apparent, for instance, in the complex time-courses of short term synaptic dynamics. Here, we suggest that

concentration codes for the uncertainty about phase – highly concentrated spiking represents high certainty about the mean phase in the cycle.

One might wonder whether uncertainty is actually important for the cases of oscillatory processing that have been identified. One key computation for spiking oscillatory networks is memory retrieval [3, 4]. Although it is not often viewed this way, memory retrieval is a genuinely probabilistic task [13, 14], with the complete answer to a retrieval query *not* being a single memory pattern, but rather a *distribution* over memory patterns. This is because at the time of the query the memory device only has access to incomplete information regarding the memory trace that needs to be recalled. Most importantly, the way memory traces are stored in the synaptic weight matrix implies a data lossy compression algorithm, and therefore the original patterns cannot be decompressed at retrieval with absolute certainty.

In this paper, we first describe how oscillatory structures can use all three activity characteristics at their disposal to represent two pieces of information and two forms of uncertainty (Section 2). We then suggest that this representational scheme is appropriate as a model of uncertainty-aware probabilistic recall in CA3. We derive the recurrent neural network dynamics that manipulate these firing characteristics such that by the end of the retrieval process neurons represent a good approximation of the posterior distribution over memory patterns given the information in the recall cue and in the synaptic weights between neurons (Section 3). We show in numerical simulations that the derived dynamics lead to competent memory retrieval, supplemented by uncertainty signals that are predictive of retrieval errors (Section 4).

## 2   Representation

**Single cell**   The heart of our proposal is a suggestion for how to interpret the activity of a single neuron in a single oscillatory cycle (such as a theta-cycle in the hippocampus) as representing a probability distribution. This is a significant extension of standard work on single-neuron representations of probability [12]. We consider a distribution over two random variables, $z \in \{0, 1\}$, a Bernoulli variable (for the case of memory, representing the participation of the neuron in the memory pattern), and $x \in [0, T)$, where $T$ is the period of the underlying oscillation, a real valued phase variable (representing an analog quantity associated with that neuron if it participates in that pattern). This distribution is based on three quantities associated with the neuron's activity (figure 1A):

$r$   the *number* of spikes in a cycle,

$\phi$   the circular *mean phase* of those spikes, under the assumption that there is at least one spike,

$c$   the *concentration* of the spikes (mean resultant length of their phases, [15]), which measures how tightly clustered they are about $\phi$

In keeping with conventional single-neuron models, we treat $r$, via a (monotonically increasing) probabilistic activation function $0 \leq \rho(r) \leq 1$, as describing the probability that $z = 1$ (figure 1B), so the distribution is $q(z; r) = \rho(r)^z (1 - \rho(r))^{1-z}$. We treat the implied distribution over the true phase $x$ as being conditional on $z$. If $z = 0$, then the phase is undefined. However, if $z = 1$, then the distribution over $x$ is a *mixture* of $q_\sqcap(x)$, a uniform distribution on $[0, T)$, and a narrow, quasi-delta, distribution $q_\perp(x; \phi)$ (of width $\epsilon \ll T$) around the mean firing phase ($\phi$) of the spikes. The mixing proportion in this case is determined by a (monotonically increasing) function $0 \leq \gamma(c) \leq 1$ of the concentration of the spikes. In total:

$$q(x, z; \phi, c, r) = [\rho(r) [\gamma(c) q_\perp(x; \phi) + (1 - \gamma(c)) q_\sqcap(x)]]^z (1 - \rho(r))^{1-z} \qquad (1)$$

as shown in figure 1C. The marginal confidence in $\phi$ being correct is thus $\lambda(c, r) = \gamma(c) \cdot \rho(r)$, which we call 'burst strength'. We can rewrite equation 1 in a more convenient form:

$$q(x, z; \phi, c, r) = [\lambda(c, r) q_\perp(x; \phi) + (\rho(r) - \lambda(c, r)) q_\sqcap(x)]^z (1 - \rho(r))^{1-z} \qquad (2)$$

**Population**   In the case of a population of neurons, the complexity of representing a full joint distribution $P[\mathbf{x}, \mathbf{z}]$ over random variables $\mathbf{x} = \{x_i\}, \mathbf{z} = \{z_i\}$ associated with each neuron $i$ grows exponentially with the number of neurons $N$. The natural alternative is to consider an approximation in which neurons make *independent* contributions, with marginals as in equation 2. The joint

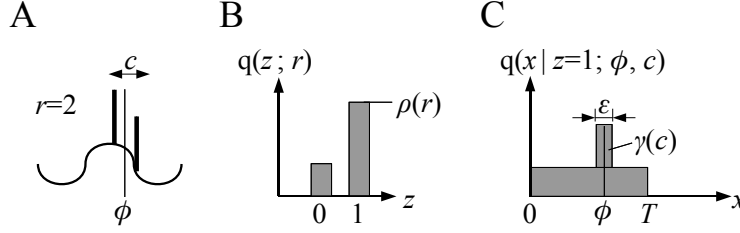

A

B    q(z ; r)

C    q(x | z=1; φ, c)

Figure 1: Representing uncertainty. *A)* A neuron's firing times during a period $[0, T)$ are described by three parameters: $r$, the number of spikes; $\phi$ the mean phase of those spikes; and $c$, the phase concentration. *B)* The firing rate $r$ determines the probability $\rho(r)$ that a Bernoulli variable associated with the unit takes the value $z = 1$. *C) If* $z = 1$, then $\phi$ and $c$ jointly define a distribution over phase which is a mixture (weighted by $\gamma(c)$) of a distribution peaked at $\phi$ and a uniform distribution.

distribution is then

$$Q\left(\mathbf{x}, \mathbf{z}; \phi, \mathbf{c}, \mathbf{r}\right) = \prod_i q\left(x_i, z_i; \phi_i, c_i, r_i\right) \tag{3}$$

whose complexity scales linearly with $N$.

**Dynamics**  When the actual distribution P the population has to represent lies outside the class of representable distributions Q in equation 3 with independent marginals, a key computational step is to find activity parameters $\phi, \mathbf{c}, \mathbf{r}$ for the neurons that make Q as close to P as possible. One way to formalize the discrepancy between the two distributions is the KL-divergence

$$\mathcal{F}\left(\phi, \mathbf{c}, \mathbf{r}\right) = \mathrm{KL}\left[Q\left(\mathbf{x}, \mathbf{z}; \phi, \mathbf{c}, \mathbf{r}\right) \parallel P\left(\mathbf{x}, \mathbf{z}\right)\right] \tag{4}$$

Minimizing this by gradient descent

$$\tau \frac{d\phi_i}{dt} = -\frac{\partial}{\partial \phi_i} \mathcal{F}\left(\phi, \mathbf{c}, \mathbf{r}\right) \qquad \tau \frac{dc_i}{dt} = -\frac{\partial}{\partial c_i} \mathcal{F}\left(\phi, \mathbf{c}, \mathbf{r}\right) \qquad \tau \frac{dr_i}{dt} = -\frac{\partial}{\partial r_i} \mathcal{F}\left(\phi, \mathbf{c}, \mathbf{r}\right) \tag{5}$$

defines dynamics for the evolution of the parameters. In general, this couples the activities of neurons, defining recurrent interactions within the network.[1]

We have thus suggested a general representational framework, in which the specification of a computational task amounts to defining a P distribution which the network should represent as best as possible. Equation 5 then defines the dynamics of the interaction between the neurons that optimizes the network's approximation.

## 3   CA3 memory

One of the most widely considered tasks that recurrent neural networks need to solve is that of autoassociative memory storage and retrieval. Moreover, hippocampal area CA3, which is thought to play a key role in memory processing, exhibits oscillatory dynamics in which firing phases are known to play an important functional role. It is therefore an ideal testbed for our theory.

We characterize the activity in CA3 neurons during recall as representing the probability distribution over memories being recalled. Treating storage from a statistical perspective, we use Bayes rule to define a posterior distribution over the memory pattern implied by a noisy and impartial cue. This distribution is represented approximately by the activities $\phi_i, r_i, c_i$ of the neurons in the network as in equation 3. Recurrent dynamics among the neurons as in equation 5 find appropriate values of these parameters, and model network interactions during recall in CA3.

**Storage**  We consider CA3 as storing patterns in which some neurons are quiet ($z_i^m = 0$, for the $i$th neuron in the $m$th pattern); and other neurons are active ($z_i^m = 1$); their activity then defining

a firing phase ($x_m^i \in [0, T)$, where $T$ is the period of the population oscillation. $M$ such memory traces, each drawn from an (*iid*) prior distribution,

$$\mathrm{P}\left[\mathbf{x}, \mathbf{z}\right] = \prod_i \left[p_z \mathrm{P}\left(x_i\right)\right]^{z_i} \left(1 - p_z\right)^{1-z_i}, \tag{6}$$

(where $p_z$ is the prior probability of firing in a memory pattern; $\mathrm{P}\left(x\right)$ is the prior distribution for firing phases) are stored locally and additively in the recurrent synaptic weight matrix of a network of $N$ neurons, $\mathbf{W}$, according to learning rule $\Omega$:

$$W_{ij} = \sum_{m=1}^{M} z_i^m z_j^m \, \Omega\left(x_i^m, x_j^m\right) \text{ for } i \neq j, \text{ and } W_{ii} = 0 \tag{7}$$

We assume that $\Omega$ is Töplitz and periodic in $T$, and either symmetric or anti-symmetric: $\Omega\left(x_1, x_2\right) = \Omega\left(x_1 - x_2\right) = \Omega\left(x_1 - x_2 \bmod T\right) = \pm\Omega\left(x_2 - x_1\right)$.

**Posterior for memory recall**   Following [14, 4], we characterize retrieval in terms of the posterior distribution over $\mathbf{x}, \mathbf{z}$ given three sources of information: a recall cue $(\tilde{\mathbf{x}}, \tilde{\mathbf{z}})$, the synaptic weight matrix, and the prior over the memories. Under some basic independence assumptions, this factorizes into three terms

$$\mathrm{P}\left[\mathbf{x}, \mathbf{z} \mid \tilde{\mathbf{x}}, \tilde{\mathbf{z}}, \mathbf{W}\right] \propto \mathrm{P}\left[\mathbf{x}, \mathbf{z}\right] \cdot \mathrm{P}\left[\tilde{\mathbf{x}}, \tilde{\mathbf{z}} \mid \mathbf{x}, \mathbf{z}\right] \cdot \mathrm{P}\left[\mathbf{W} \mid \mathbf{x}, \mathbf{z}\right] \tag{8}$$

The first term is the prior (equation 6). The second term is the likelihood of receiving noisy or partial recall cue $(\tilde{\mathbf{x}}, \tilde{\mathbf{z}})$ if the true pattern to be recalled was $(\mathbf{x}, \mathbf{z})$:

$$\mathrm{P}\left[\tilde{\mathbf{x}}, \tilde{\mathbf{z}} \mid \mathbf{x}, \mathbf{z}\right] = \prod_i \left[\left(\eta_1 \, \tilde{\mathrm{P}}_1\left(\tilde{x}_i \mid x_i\right)\right)^{\tilde{z}_i} \left(1 - \eta_1\right)^{1-\tilde{z}_i}\right]^{z_i} \left[\left(\left(1 - \eta_0\right) \tilde{\mathrm{P}}_0\left(\tilde{x}_i\right)\right)^{\tilde{z}_i} \eta_0^{1-\tilde{z}_i}\right]^{1-z_i} \tag{9}$$

where $\eta_1 = \mathrm{P}\left[\tilde{z} = 1 \mid z = 1\right]$ and $\eta_0 = \mathrm{P}\left[\tilde{z} = 0 \mid z = 0\right]$ are the probabilities of the presence or absence of a spike in the input given the presence or absence of a spike in the memory to be recalled, $\tilde{\mathrm{P}}_1\left(\tilde{x} \mid x\right)$ and $\tilde{\mathrm{P}}_0\left(\tilde{x}\right)$ are distributions of the phase of an input spike if there was or was not a spike in the memory to be recalled.

The last term in equation 8 is the likelihood that weight matrix $\mathbf{W}$ arose from $M$ patterns *including* $(\mathbf{x}, \mathbf{z})$. Making a factorized approximation $\mathrm{P}\left[\mathbf{W} \mid \mathbf{x}, \mathbf{z}\right] \simeq \left(\prod_{i, j \neq i} \mathrm{P}\left[W_{ij} \mid x_i, z_i, x_j, z_j\right]\right)^{1/2}$. Since the learning rule is additive and memory traces are drawn *iid*, the likelihood of a synaptic weight is approximately Gaussian for large $M$, with a quadratic log-likelihood [4]:

$$\log \mathrm{P}\left[W_{ij} \mid x_i, z_i, x_j, z_j\right] \overset{+c}{=} \frac{z_i z_j}{\sigma_W^2} \left[\left(W_{ij} - \mu_W\right) \Omega\left(x_i, x_j\right) - \frac{1}{2}\Omega^2\left(x_i, x_j\right)\right] \tag{10}$$

where $\mu_W$ and $\sigma_W^2$ are the mean and variance of the distribution of synaptic weights after storing $M - 1$ random memory traces ($\mu_W = 0$ for antisymmetric $\Omega$).

**Dynamics for memory recall**   Plugging the posterior from equation 8 to the general dynamics equation 5 yields the neuronal update rules that will be appropriate for uncertainty-aware memory recall, and which we treat as a model of recurrent dynamics in CA3.

We give the exact formulæ for the dynamics in the supplementary material. They can be shown to couple together the various activity parameters of the neurons in appropriate ways, for instance weighting changes to $\phi_i$ for neuron $i$ according to the burst strength of its presynaptic inputs, and increasing the concentration when the log posterior of the firing phase of the neuron, given that it should fire, $\log \mathrm{P}[\phi_i | z_i = 1, \tilde{\mathbf{x}}, \tilde{\mathbf{z}}, \mathbf{W}]$, is greater than the average of the log posterior.

These dynamics generalize, and thus inherit, some of the characteristics of the purely phase-based network suggested in [4]. This means that they also inherit the match with physiologically-measured phase response curves (PRCs) from *in vitro* CA3 neurons that were measured to test this suggestion [16]. The key difference here is that we expect the magnitude (though not the shape) of the influence of a presynaptic neuron on the phase of a postsynaptic one to scale with its rate, for high concentration. Preliminary *in vitro* results show that PRCs recorded in response to burst stimulation are not qualitatively different from PRCs induced by single spikes; however, it remains to be seen if their magnitude scales in the way implied by the dynamics here.

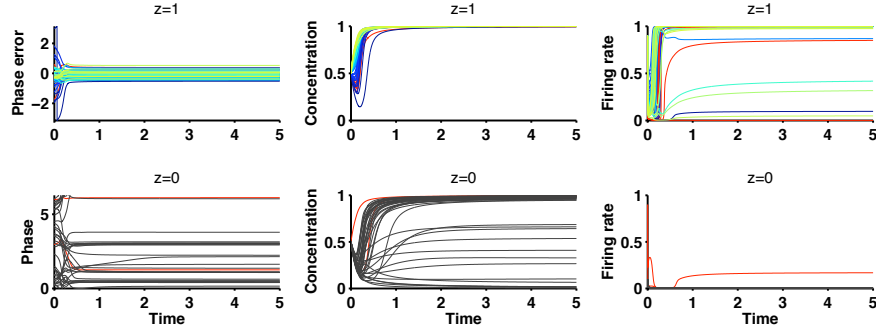

Figure 2: A single retrieval trial in the network. Time evolution of firing phases *(left panels)*, concentrations *(middle panels)*, and rates *(right panels)* of neurons that should *(top row)* or should not *(bottom row)* participate in the memory pattern being retrieved. Note that firing phases in the top row are plotted as a difference from the stored firing phases so that $\phi = 0$ means perfect retrieval. Color code shows precision *(blue: low, yellow: high)* of the phase of the input to neurons, with *red lines* showing cells receiving incorrect input rate.

## 4    Simulations

Figure 2 shows the course of recall in the full network (with $N = 100$ neurons, and 10 stored patterns with $p_z = 0.5$). For didactic convenience, we consider the case that the noise in the phase input was varied systematically for different neurons within a recall cue (a fact known to the network, *ie* incorporated into its dynamics), so that it is possible to see how the differential certainty evolves over the course of the network's dynamics. The *top left panel* shows that neurons that should fire in the memory trace (*ie* for which $z = 1$) quickly converge on their correct phase, and that this convergence usually takes a longer time for neurons receiving more uncertain input. This is paralleled by the way their firing concentrations change *(top middle panel)*: neurons with reliable input immediately increase their concentrations from the initial $\gamma(c) = 0.5$ value to $\gamma(c) = 1$, while for those having more unreliable input it takes a longer time to build up confidence about their firing phases (and by the time they become confident their phases are indeed correct). Neurons that should not fire ($z = 0$) build up their confidence even more slowly, more often remain fairly uncertain or only moderately certain about their firing phases, as expressed by their concentrations *(middle bottom panel)* – quite righteously. Finally, since the firing rate input to the network is correct 90%, most neurons that should or should not fire do or do not fire, respectively, with maximal certainty about their rate *(top and bottom right panels)*.

Various other metrics are important for providing insight into the operation of the network. In particular, we may expect there to be a relationship between the *actual* error in the phase of firing of the neurons recalled by the memory, and the firing rates and concentrations (in the form of burst strengths) of the associated neurons themselves. Neurons which are erring should whisper rather than shout. Figure 3A shows just this for the network. Here, we have sorted the neurons according to their burst strengths $\lambda$, and plotted histograms of errors in firing phase for each group. The lower the burst strength, the more likely are large errors – at least to an approximation. A similar relationship exists between recalled (analogue) and stored (binary) firing rates, where extreme values of the recalled firing rate indicate that the stored firing rate was 0 or 1 with higher certainty (Figure 3B).

Figure 3C shows the results of a related analysis of experimental data kindly donated by Francesco Battaglia. He recorded neurons in hippocampal area CA1 (not CA3, although we may hope for some similar properties) whilst rats were shuttling on a linear track for food reward. CA1 neurons have place fields – locations in the environment where they respond with spikes – and the phases of these spikes relative to the ongoing theta oscillation in the hippocampus are also known to convey information about location in space [5]. To create the plot, we first selected epochs with high-quality and high power theta activity in the hippocampus (to ensure that phase is well estimated). We then computed the mean firing phase within the theta cycle, $\phi$, of each neuron as a function of the location of the rat, separately for each visit to the same location. We assumed that the 'true' phase $x$ a neuron should recall at a given location is the average of these phases across different visits. We

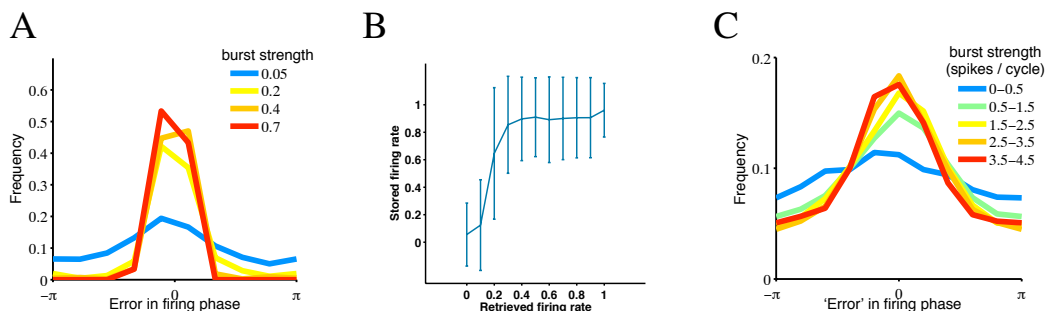

Figure 3: Uncertainty signals are predictive of the error a cell is making both in simulation *(A,B)*, and as recorded from behaving animals *(C)*. Burst strength signals overall uncertainty about and thus predicts error in mean firing phase *(A,C)*, while graded firing rates signal certainty about whether to fire or not *(B)*.

then evaluated the error a neuron was making at a given location on a given visit as the difference between its $\phi$ in that trial at that location and the 'true' phase $x$ associated with that location. This allowed us to compute statistics of the error in phase as a function of the burst strength. The curves in the figure show that, as for the simulation, burst strength is at least partly inversely correlated with actual phase error, defined in terms of the overall activity in the population. Of course, this does not constitute a proof of our representational theory.

One further way to evaluate the memory is to compare it to two existing associative memories that have previously been studied, and can be seen as special cases. On one hand, our memory adds the dimension of phase to the uncertainty-aware rate-based memory that Sommer & Dayan [14] studied. This memory made a somewhat similar variational approximation, but, as for the mean-field Boltzmann machine [17], only involving $r$ and $\rho(r)$ and no phases.

On the other hand, the memory device can be seen as adding the dimension of rate to the phase-based memory that Lengyel & Dayan [4] treated. Note, however, that although this phase-based network used superficially similar probabilistic principles to the one we have developed here, in fact it did not operate according to uncertainty, since it made the key simplification that all neurons participate in all memories, and that they also fire exactly one spike on every cycle during recall. This restricted the dynamics of that network to perform maximum *a posteriori* (MAP) inference to find the single recalled pattern of activity that best accommodated the probabilistic constraints of the cue, the prior and the synaptic weights, rather than being able to work in the richer space of probabilistic recall of the dynamics we are suggesting here.

Given these roots, we can follow the logic in figure 4 and compare the performance of our memory with these precursors in the cases for which they are designed. For instance, to compare with the rate-based network, we construct memories which include phase information. During recall, we present cues with relatively accurate rates, but relatively inaccurate phases, and evaluate the extent to which the network is perturbed by the presence of the phases (which, of course, it has to store in the single set of synaptic weights). Figure 4A shows exactly this comparison. Here, a relatively small network ($N = 100$) was used, with memories that are dense ($p_z = 0.5$), and it is therefore a stringent test of the storage capacity. Performance is evaluated by calculating the average error made in recalled firing rates).

In the figure, the two blue curves are for the full model (with the phase information in the input being relatively unreliable, its circular concentration parameter distributed uniformly between 0.1 and 10 across cells); the two yellow curves are for a network with only rates (which is similar to that described, but not simulated, by Sommer & Dayan [14]). Exactly the same rate information is provided to all networks, and is 10% inaccurate (a degree known to the dynamics in the form of $\eta_0$ and $\eta_1$). The two flat dashed lines show the performance in the case that there are no recurrent synaptic weights at all. This is an important control, since we are potentially presenting substantial information in the cues themselves. The two solid curves show that the full model tracks the reduced, rate-based, model almost perfectly until the performance totally breaks down. This shows that the phase information, and the existence of phase uncertainty and processing during recall, does not

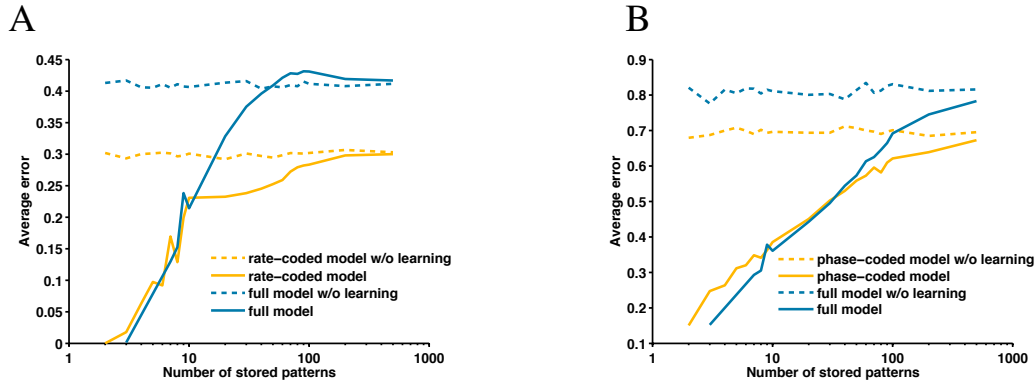

Figure 4: Recall performance compared with a rate-only network *(A)* and a phase-only network *(B)*. The full model *(blue lines)* performs just as well as the reduced 'specialist' models *(yellow lines)* in comparable circumstances (when the information provided to the networks in the dimension they shared is exactly the same). All models *(solid lines)* outperform the standard control of using the input and the prior alone *(dashed lines)*.

corrupt the network's capacity to recall rates. Given its small size, the network is quite competent as an auto-associator.

Figure 4B shows a similar comparison between this network and a network that only has to deal with uncertainty in firing phases but not in rates. Again, its performance at recalling phase, given uncertain and noisy phase cues, but good rate-cues, is exactly on a par with the pure, phase-based network. Further, the average errors are only modest, so the capacity of the network for storing analog phases is also impressive.

## 5   Discussion

We have considered an interpretation of the activities of neurons in oscillating structures such as area CA3 of the hippocampus as representing distributions over two underlying quantities, one binary and one analogue. We also showed how this representational capacity can be used to excellent effect in the key, uncertainty-sensitive computation of memory recall, an operation in which CA3 is known to be involved. The resulting network model of CA3 encompasses critical aspects of its physiological properties, notably information-bearing firing rates and phases. Further, since it generalizes earlier theories of purely phase-based memories, this model is also consistent with the measured phase response curves of CA3 neurons, which characterize their actual dynamical interactions.

Various aspects of this new theory are amenable to experimental investigation. First, the full dynamics (see the supplementary material) imply that firing rate and firing phase should be coupled together both pre-synpatically, in terms of the influence of timed input spikes, and post-synaptically, in terms of how changes in the activity of a neuron should depend on its own activity. *In vitro* experiments along the lines of those carried out before [16], in which we have precise experimental control over pre- and post-synaptic activity can be used to test these predictions. Further, making the sort of assumptions that underlie figure 3C, we can use data from awake behaving rats to see if the gross statistics of the changes in the activity of the neurons fit the expectations licensed by the theory.

From a computational perspective, we have demonstrated that the network is a highly competent associative memory, correctly recalling both binary and analog information, along with certainty about it, and degrading gracefully in the face of overload. In fact, compared with the representation of other analogue quantities (such as the orientation of a visually preseted bar), analogue memory actually poses a particularly tough problem for the representation of uncertainty. This is because for variables like orientation, a whole population is treated as being devoted to the representation of the distribution of a single scalar value. By contrast, for analogue memory, each neuron has an independent analogue value, and so the dimensionality of the distribution scales with the number of neurons involved. This extra representational power comes from the ability of neurons to distribute

their spikes within a cycle to indicate their uncertainty about phase (using the dimension of time in just the same way that distributional population codes [12] used the dimension of neural space).

This dimension for representing analogue uncertainty is coupled to that of the firing rate for representing binary uncertainty, since neurons have to fire multiple times in a cycle to have a measurable lack of concentration. However, this coupling is exactly appropriate given the form of the distribution assumed in equation 2, since weakly firing neurons express only weak certainty about phase in any case. In fact, it is conceivable that we could combine a different model for the firing rate uncertainty with this model for analogue uncertainty, if, for instance, it is found that neuronal firing rates covary in ways that are not anticipated from equation 2.

Finally, the most important direction for future work is understanding the uncertainty-sensitive coupling between multiple oscillating memories, where the oscillations, though dynamically coordinated, need not have the same frequencies. Exactly this seems to characterize the interaction between the hippocampus and the necortex during both consolidation and retrieval [18, 19].

**Acknowledgments**

Funding from the Gatsby Charitable Foundation. We are very grateful to Francesco Battaglia for allowing us to use his data to produce figure 3C, and to him, and Ole Paulsen and Jeehyun Kwag for very helpful discussions.

**References**

[1] Szalisznyó K, Érdi P. In The Handbook of Brain Theory and Neural Networks, 533, 2003.
[2] Hopfield JJ. Nature 376:33, 1995.
[3] Yoshioka M. Physical Review E 65, 2001.
[4] Lengyel M, Dayan P. In Advances in Neural Information Processing Systems 17, 769, Cambridge, MA, 2005. MIT Press.
[5] O'Keefe J, Recce ML. Hippocampus 3:317, 1993.
[6] Huxter J, et al. Nature 425:828, 2003.
[7] Ernst M, Banks M. Nature 415:429, 2002.
[8] Körding K, Wolpert D. Nature 427:244, 2004.
[9] Gold JI, Shadlen MN. Neuron 36:299, 2002.
[10] Hinton G. Neural Comput 1:143, 1990.
[11] Peterson C, Anderson J. Complex Systems 1:995, 1987.
[12] Pouget A, et al. Annu Rev Neurosci 26:381, 2003.
[13] MacKay DJC. In Maximum Entropy and Bayesian Methods, Laramie, 1990, 237, 1991.
[14] Sommer FT, Dayan P. IEEE Trans Neural Netw 9:705, 1998.
[15] Fisher NI. Statistical analysis of circular data. Cambridge University Press, 1995.
[16] Lengyel M, et al. Nat Neurosci 8:1677, 2005.
[17] Dayan P, Abbott LF. Theoretical Neuroscience. MIT Press, 2001.
[18] Siapas AG, Wilson MA. Neuron 21:1123, 1998.
[19] Jones M, Wilson M. PLoS Biol 3:e402, 2005.

## Footnotes

[1]Of course, the firing rate is really an integer variable, since it is an actual number of spikes per cycle. For simplicity, in the simulations below, we considered real-valued firing rates – an important next step is to drop this assumption.
